# Collective Graphical Models

**Daniel Sheldon**
Oregon State University
sheldon@eecs.oregonstate.edu

**Thomas G. Dietterich**
Oregon State University
tgd@eecs.oregonstate.edu

## Abstract

There are many settings in which we wish to fit a model of the behavior of in-
dividuals but where our data consist only of aggregate information (counts or
low-dimensional contingency tables). This paper introduces *Collective Graphi-
cal Models*—a framework for modeling and probabilistic inference that operates
directly on the sufficient statistics of the individual model. We derive a highly-
efficient Gibbs sampling algorithm for sampling from the posterior distribution
of the sufficient statistics conditioned on noisy aggregate observations, prove its
correctness, and demonstrate its effectiveness experimentally.

## 1   Introduction

In fields such as ecology, marketing, and the social sciences, data about identifiable individuals is
rarely available, either because of privacy issues or because of the difficulty of tracking individuals
over time. Far more readily available are aggregated data in the form of counts or low-dimensional
contingency tables. Despite the fact that only aggregated data are available, researchers often seek
to build models and test hypotheses about individual behavior. One way to build a model connecting
individual-level behavior to aggregate data is to explicitly model each individual in the population,
together with the aggregation mechanism that yields the observed data.

However, with large populations it is infeasible to reason about each individual. Luckily, for many
purposes it is also unnecessary. To fit a probabilistic model of individual behavior, we only need
the sufficient statistics of that model. This paper introduces a formalism in which one starts with a
graphical model describing the behavior of individuals, and then derives a new graphical model —
the *Collective Graphical Model (CGM)* — on the sufficient statistics of a population drawn from
that model. Remarkably, the CGM has a structure similar to that of the original model.

This paper is devoted to the problem of inference in CGMs, where the goal is to calculate conditional
probabilities over the sufficient statistics given partial observations made at the population level. We
consider both an exact observation model where subtables of the sufficient statistics are observed
directly, and a noisy observation model where these counts are corrupted. A primary application is
learning: for example, computing the expected value of the sufficient statistics comprises the "E"
step of an EM algorithm for learning the individual model from aggregate data.

**Main concepts.** The ideas behind CGMs are best illustrated by an example. Figure 1(a) shows the
graphical model plate notation for the bird migration model from [1, 2], in which birds transition
stochastically among a discrete set of locations (say, grid cells on a map) according to a Markov
chain (the *individual model*). The variable $X_t^m$ denotes the location of the $m$th bird at time $t$, and
birds are independent and identically distributed. This model gives an explicit way to reason about
the interplay between individual-level behavior (inside the plate) and aggregate data. Suppose, for
example, that very accurate surveys reveal the number of birds $n_t(i)$ in each location $i$ at each time
$t$, and these numbers are collected into a single vector $\mathbf{n}_t$ for each time step. Then, for example, one
can compute the likelihood of the survey data given parameters *of the individual model* by summing
out the individual variables. However, this is highly impractical: if our map has $L$ grid cells, then
the variable elimination algorithm run on this model would instantiate tabular potentials of size $L^M$.

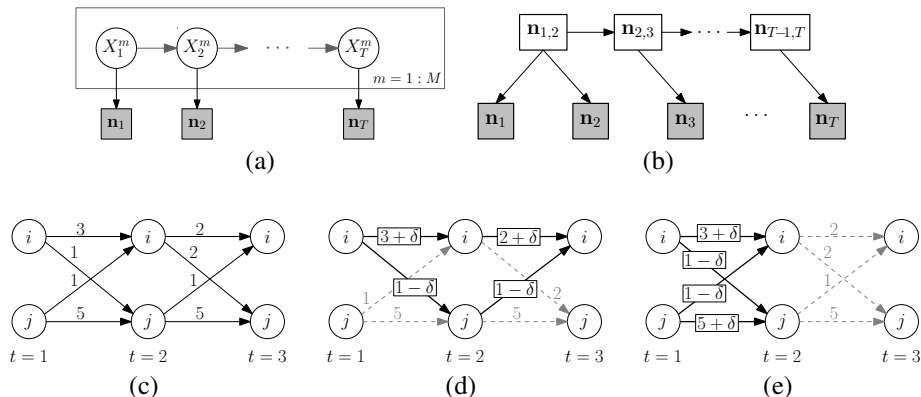

Figure 1: Collective graphical model of bird migration: (a) replicates of individual model connected to population-level observations, (b) CGM after marginalizing away individuals, (c) trellis graph on locations $\{i, j\}$ for $T = 3, M = 10$; numbers on edges indicate flow amounts, (d) a *degree-one* cycle; flows remain non-negative for $\delta \in \{-3, \ldots, 1\}$, (e) a *degree-two* cycle; flows remain non-negative for $\delta \in \{-2, \ldots, 1\}$.

Figure 1(b) shows the CGM for this model, which we obtain by *analytically* marginalizing away the individual variables to get a new model on their sufficient statistics, which are the tables $\mathbf{n}_{t,t+1}$ with entries $n_{t,t+1}(i, j)$ equaling the number of birds that fly from $i$ to $j$ from time $t$ to $t + 1$. A much better inference approach would be to conduct variable elimination or message passing directly in the CGM. However, this would still instantiate potentials that are much too big for realistic problems due to the huge state space: e.g., there are $\binom{M+L^2-1}{L^2-1} = O(M^{L^2-1})$ possible values for the table $\mathbf{n}_{t,t+1}$.

Instead, we will perform approximate inference using MCMC. Here, we are faced with yet another challenge: the CGM has hard constraints encoded into its distribution, and our MCMC moves must preserve these constraints yet still connect the state space. To understand this, observe that the hidden variables in this example comprise a flow of $M$ units through the *trellis graph* of the Markov chain, with the interpretation that $n_{t,t+1}(i, j)$ birds "flow" along edge $(i, j)$ at time $t$ (see Figure 1(c) and [1]). The constraints are that (1) flow is conserved at each trellis node, and (2) the number of birds that enter location $i$ at time $t$ equals the observed number $n_t(i)$. (In the case of noisy or partial observations, the latter constraint may not be present.)

How can we design a set of moves that connect any two $M$-unit flows while preserving these constraints? The answer is to make moves that send flow around cycles. Cycles of the form illustrated in Figure 1(d) preserve flow conservation but change the amount of flow through some trellis nodes. Cycles of the form in Figure 1(e) preserve both constraints. One can show by graph-theoretic arguments that moves of these two general classes are enough to connect any two flows.

This gives us the skeleton of an ergodic MCMC sampler: starting with a feasible flow, select cycles from these two classes uniformly at random and propose moves that send $\delta$ units of flow around the cycle. There is one unassuming but crucially important final question: how to select $\delta$? The following is a form of Gibbs sampler: from all values that preserve non-negativity, select $\delta$ with probability proportional to that of the new flow. Such moves are always accepted. Remarkably, even though $\delta$ may take on as many as $M$ different values, the resulting distribution over $\delta$ has an extremely tractable form — either binomial or hypergeometric — and thus it is possible to select $\delta$ in constant time, so we can make *very large moves in time independent of the population size*.

**Contributions.** This paper formally develops these concepts in a way that generalizes the construction of Figure 1 to allow arbitrary graphical models inside the plate, and a more general observation model that includes both noisy observations and observations involving multiple variables. We develop an efficient Gibbs sampler to conduct inference in CGMs that builds on existing work for conducting *exact tests* in contingency tables and makes several novel technical contributions. Foremost is the analysis of the distribution over the move size $\delta$, which we show to be a discrete univariate distribution that generalizes both the binomial and hypergeometric distributions. In particular, we prove that it is always *log-concave* [3], so it can be sampled in constant expected running time. We

show empirically that resulting inference algorithm runs in time that is *independent of the population size*, and is dramatically faster than alternate approaches.

**Related Work.** The bird migration model of [1, 2] is a special case of CGMs where the individual model is a Markov chain and observations are made for single variables only. That work considered only maximum a posteriori (MAP) inference; the method of this paper could be used for learning in that application. Sampling methods for exact tests in contingency tables (e.g. [4]) generate tables with the same sufficient statistics as an observed table. Our work differs in that our observations are *not* sufficient, and we are sampling the sufficient statistics instead of the complete contingency table. Diaconis and Sturmfels [5] broadly introduced the concept of *Markov bases*, which are sets of moves that connect the state space when sampling from conditional distributions by MCMC. We construct a Markov basis in Section 3.1 based on work of Dobra [6]. Lauritzen [7] discusses the problem of exact tests in nested decomposable models, a setup that is similar to ours. Inference in CGMs can be viewed as a form of *lifted inference* [8–12]. The counting arguments used to derive the CGM distribution (see below) are similar to the operations of *counting elimination* [9] and *counting conversion* [10] used in exact lifted inference algorithms for first-order probabilistic models. However, those algorithms do not replicate the CGM construction when applied to a first-order representation of the underlying population model. For example, when applied to the bird migration model, the C-FOVE algorithm of Milch et al. [10] cannot introduce contingency tables over pairs of variables $(X_t, X_{t+1})$ as required to represent the sufficient statistics; it can only introduce histograms over single variables $X_t$. Apsel and Brafman [13] have recently taken a step in this direction by introducing a lifting operation to construct the Cartesian product of two first-order formulas. In the applications we are considering, exact inference (even when lifted) is intractable.

## 2   Problem Setup

Let $(X_1, X_2, \ldots, X_{|V|})$ be a set of discrete random variables indexed by the finite set $V$, where $X_v$ takes values in the set $\mathcal{X}_v$. Let $\mathbf{x} = (x_1, \ldots, x_{|V|})$ denote a joint setting for these variables from the set $\mathcal{X} = \mathcal{X}_1 \times \ldots \times \mathcal{X}_{|V|}$. For our individual model, we consider graphical models of the form:

$$p(\mathbf{x}) = \frac{1}{Z} \prod_{C \in \mathcal{C}} \phi_C(\mathbf{x}_C). \tag{1}$$

Here, $\mathcal{C}$ is the set of cliques of the independence graph, the functions $\phi_C : \mathcal{X}_C \to \mathbb{R}^+$ are *potentials*, and $Z$ is a normalization constant. For $A \subset V$, we use the notation $\mathbf{x}_A$ to indicate the sub-vector of variables with indices belonging to $A$, and use similar notation for the corresponding domain $\mathcal{X}_A$. We also assume that $p(\mathbf{x}) > 0$ for all $\mathbf{x} \in \mathcal{X}$, which is required for our sampler to be ergodic. Models that fail this restriction can be modified by adding a small positive amount to each potential.

A *collection* $\mathcal{A}$ is a set of subsets of $V$. For collections $\mathcal{A}$ and $\mathcal{B}$, define $\mathcal{A} \preceq \mathcal{B}$ to mean that each $A \in \mathcal{A}$ is contained in some $B \in \mathcal{B}$. A collection $\mathcal{A}$ is *decomposable* if there is a *junction tree* $\mathcal{T} = (\mathcal{A}, \mathcal{E}(\mathcal{T}))$ on vertex set $\mathcal{A}$ [7]. Any collection $\mathcal{A}$ can be extended to a decomposable collection $\mathcal{B}$ such that $\mathcal{A} \preceq \mathcal{B}$; this corresponds to adding fill-in edges to a graphical model.

Consider a sample $\{\mathbf{x}^{(1)}, \ldots, \mathbf{x}^{(M)}\}$ from the graphical model. A *contingency table* $\mathbf{n} = (n(i))_{i \in \mathcal{X}}$ has entries $n(i) = \sum_{m=1}^{M} I\{\mathbf{x}^{(m)} = i\}$ that count the number of times each element $i \in \mathcal{X}$ appears in the sample. We use index variables such as $i, j \in \mathcal{X}$ (instead of $\mathbf{x} \in \mathcal{X}$) to refer to cells of the contingency table, where $i = (i_1, \ldots, i_V)$ is a vector of indices and $i_A$ is the sub-vector corresponding to $A \subseteq V$. Let $\mathrm{tbl}(A)$ denote the set of all valid contingency tables on the domain $\mathcal{X}_A$. A valid table is indexed by elements $i_A \in \mathcal{X}_A$ and has non-negative integer entries. For a full table $\mathbf{n} \in \mathrm{tbl}(V)$ and $A \subseteq V$, let the *marginal table* $\mathbf{n} \downarrow A \in \mathrm{tbl}(A)$ be defined as $(\mathbf{n} \downarrow A)(i_A) = \sum_{m=1}^{M} I\{\mathbf{x}_A^{(m)} = i_A\} = \sum_{i_B \in \mathcal{X}_{V \setminus A}} n(i_A, i_B)$. When $A = \emptyset$, define $\mathbf{n} \downarrow A$ to be the scalar $M$, the grand total of the table. Write $\mathbf{n}_A \preceq \mathbf{n}_B$ to mean that $\mathbf{n}_A$ is a marginal table of $\mathbf{n}_B$ (i.e., $A \subseteq B$ and $\mathbf{n}_A = \mathbf{n}_B \downarrow A$)

Our observation model is as follows. We assume that a sample $\{\mathbf{x}^{(1)}, \ldots, \mathbf{x}^{(M)}\}$ is drawn from the individual model, resulting in a complete, but unobserved, contingency table $\mathbf{n}_V$. We then observe the marginal tables $\mathbf{n}_D = \mathbf{n}_V \downarrow D$ for each set $D$ in a collection of *observed margins* $\mathcal{D}$, which we require to be decomposable. Write this overall collection of tables as $\mathbf{n}_\mathcal{D} = \{\mathbf{n}_D\}_{D \in \mathcal{D}}$. We consider noisy observations in Section 3.3.

**Building the CGM.** In a discrete graphical model, the sufficient statistics are the contingency tables $\mathbf{n}_{\mathcal{C}} = \{\mathbf{n}_C\}_{C \in \mathcal{C}}$ over cliques. Our approach relies on the ability to derive a tractable probabilistic model for these statistics by marginalizing out the sample. If $\mathcal{C}$ is decomposable, this is possible, so let us assume that $\mathcal{C}$ has a junction tree $\mathcal{T}_{\mathcal{C}}$ (if not, fill-in edges must be added to the original model). Let $\boldsymbol{\mu}_C$ be the table of marginal probabilities for clique $C$ (i.e. $\mu_C(i_C) = \Pr(X_C = i_C)$). Let $\mathcal{S}$ be the collection of separators of $\mathcal{T}_{\mathcal{C}}$ (with repetition if the same set appears as a separator multiple times) and let $\mathbf{n}_S$ and $\boldsymbol{\mu}_S$ be the tables of counts and marginal probabilities for the separator $S \in \mathcal{S}$.

The distribution of $\mathbf{n}_{\mathcal{C}}$ was first derived by Sundberg [14]:

$$ p(\mathbf{n}_{\mathcal{C}}) = M! \left( \prod_{C \in \mathcal{C}} \prod_{i_C \in \mathcal{X}_C} \frac{\mu_C(i_C)^{n_C(i_C)}}{n_C(i_C)!} \right) \left( \prod_{S \in \mathcal{S}} \prod_{i_S \in \mathcal{X}_S} \frac{\mu_S(i_S)^{n_S(i_S)}}{n_S(i_S)!} \right)^{-1}, \qquad (2) $$

which can be understood as a product of multinomial distributions corresponding to a sampling scheme for $\mathbf{n}_{\mathcal{C}}$ (details omitted). It is this distribution that we call the collective graphical model; the parameters are the marginal probabilities of the individual model. To understand the conditional distribution given the observations, let us further assume that $\mathcal{D} \preceq \mathcal{C}$ (if not, add additional fill-in edges for variables that co-occur within $\mathcal{D}$), so that each observed table is determined by some clique table. Write $\mathbf{n}_{\mathcal{D}} \preceq \mathbf{n}_{\mathcal{C}}$ to express the condition that the tables $\mathbf{n}_{\mathcal{C}}$ produce observations $\mathbf{n}_{\mathcal{D}}$: formally, this means that $\mathcal{D} \preceq \mathcal{C}$ and that $D \subseteq C$ implies that $\mathbf{n}_D \preceq \mathbf{n}_C$. Let $I\{\cdot\}$ be an indicator variable. Then

$$ p(\mathbf{n}_{\mathcal{C}} \mid \mathbf{n}_{\mathcal{D}}) \propto p(\mathbf{n}_{\mathcal{C}}, \mathbf{n}_{\mathcal{D}}) = p(\mathbf{n}_{\mathcal{C}})I\{\mathbf{n}_{\mathcal{D}} \preceq \mathbf{n}_{\mathcal{C}}\}. \qquad (3) $$

In general, the number of contingency tables over small sets of variables leads to huge state spaces that prohibit exact inference schemes using (2) and (3). Thus, our approach is based on Gibbs sampling. However, there are two constraints that significanlty complicate sampling. First, the clique tables must match the observations (i.e., $\mathbf{n}_{\mathcal{D}} \preceq \mathbf{n}_{\mathcal{C}}$). Second, implicit in (2) is the constraint that the tables $\mathbf{n}_{\mathcal{C}}$ must be *consistent* in the sense that they are the sufficient statistics of some sample, otherwise $p(\mathbf{n}_{\mathcal{C}}) = 0$.

**Definition 1.** *Refer to the set of contingency tables* $\mathbf{n}_{\mathcal{A}} = \{\mathbf{n}_A\}_{A \in \mathcal{A}}$ *as a* configuration. *A configuration is* (globally) consistent *if there exists* $\mathbf{n}_V \in tbl(V)$ *such that* $\mathbf{n}_A = \mathbf{n}_V \downarrow A$ *for all* $A \in \mathcal{A}$.

Consistency requires, for example, that any two tables must agree on their common marginal, which yields the flow conservation constraints in the bird migration model. Table entries must be carefully updated in concert to maintain these constraints. A full discussion follows.

## 3 Inference

Our goal is to develop a sampler for $p(\mathbf{n}_{\mathcal{C}} \mid \mathbf{n}_{\mathcal{D}})$ given the observed tables $\mathbf{n}_{\mathcal{D}}$. We assume that the CGM specified in Equations (1) and (2) satisfies $\mathcal{D} \preceq \mathcal{C}$, and that the configuration $\mathbf{n}_{\mathcal{D}}$ is consistent.

**Initialization.** The first step is to construct a valid initial value for $\mathbf{n}_{\mathcal{C}}$, which must be a globally consistent configuration satisfying $\mathbf{n}_{\mathcal{D}} \preceq \mathbf{n}_{\mathcal{C}}$. Doing so without instantiating huge intermediate tables requires a careful sequence of operations on the two junction trees $\mathcal{T}_{\mathcal{C}}$ and $\mathcal{T}_{\mathcal{D}}$. We state one key theorem, but defer the full algorithm, which is lengthy and technical, to the supplement.

**Theorem 1.** *Let* $\mathcal{A}$ *be a decomposable collection with junction tree* $\mathcal{T}_{\mathcal{A}}$. *Say that the configuration* $\mathbf{n}_{\mathcal{A}}$ *is* locally consistent *if it agrees on edges of* $\mathcal{T}_{\mathcal{A}}$, *i.e., if* $\mathbf{n}_A \downarrow S = \mathbf{n}_B \downarrow S$ *for all* $(A, B) \in \mathcal{E}(\mathcal{T}_{\mathcal{A}})$ *with* $S = A \cap B$. *If* $\mathbf{n}_{\mathcal{A}}$ *is locally consistent, then it is also globally consistent.*

In the bird migration example, Theorem 1 guarantees that preserving flow conservation is enough to maintain consistency. It is structurally equivalent to the "junction tree theorem" (e.g., [15]) which asserts that marginal probability tables $\{\boldsymbol{\mu}_A\}_{A \in \mathcal{A}}$ that are locally consistent are realizable as the marginals of some joint distribution $p(\mathbf{x})$. Like that result, Theorem 1 also has a constructive proof, which is the foundation for our initialization algorithm. However, the integrality requirements of contingency tables necessitate a different style of construction.

### 3.1 Markov Basis

The first key challenge in designing the MCMC sampler is constructing a set of moves that preserve the constraints mentioned above, yet still connect any two points in the support of the distribution. Such a set of moves is called a *Markov basis* [5].

**Definition 2.** *A set of moves $\mathcal{M}$ is a Markov basis for the set $\mathcal{F}$ if, for any two configurations $\mathbf{n}, \mathbf{n}' \in \mathcal{F}$, there is a sequence of moves $\mathbf{z}_1, \ldots, \mathbf{z}_L \in \mathcal{M}$ such that: (i) $\mathbf{n}' = \mathbf{n} + \sum_{\ell=1}^{L} \mathbf{z}_\ell$, and (ii) $\mathbf{n} + \sum_{\ell=1}^{L'} \mathbf{z}_\ell \in \mathcal{F}$ for all $L' = 1, \ldots, L - 1$.*

In our problem, the set we wish to connect is the support of $p(\mathbf{n}_\mathcal{C} \mid \mathbf{n}_\mathcal{D})$. Our positivity assumption on $p(\mathbf{x})$ implies that any consistent configuration $\mathbf{n}_\mathcal{C}$ has positive probability, and thus the support of $p(\mathbf{n}_\mathcal{C} \mid \mathbf{n}_\mathcal{D})$ is exactly the set of consistent configurations that match the observations:

$$\mathcal{F}_{\mathbf{n}_\mathcal{D}} = \{\mathbf{n}_\mathcal{C} : \mathbf{n}_\mathcal{C} \text{ is consistent and } \mathbf{n}_\mathcal{D} \preceq \mathbf{n}_\mathcal{C}\}$$

It is useful at this point to think of the configuration $\mathbf{n}_\mathcal{C}$ as a vector obtained by sorting the table entries in any consistent fashion (e.g., lexicographically first by $C \in \mathcal{C}$ and then by $i_C \in \mathcal{X}_C$). A move can be expressed as $\mathbf{n}'_\mathcal{C} = \mathbf{n}_\mathcal{C} + \mathbf{z}$ where $\mathbf{z}$ is an integer-valued vector of the same dimension as $\mathbf{n}_\mathcal{C}$ that may have negative entries.

**The Dobra Markov basis for complete tables.** Dobra [6] showed how to construct a Markov basis for moves in a *complete* contingency table given a decomposable set of margins. Specifically, let $\mathcal{A}$ be decomposable and let $\mathbf{n}_\mathcal{A}$ be consistent with $\bigcup \mathcal{A} = V$, so that each variable is part of an observed margin. Define $\mathcal{F}^*_{\mathbf{n}_\mathcal{A}} = \{\mathbf{n}_V \in \text{tbl}(V) : \mathbf{n}_\mathcal{A} \preceq \mathbf{n}_V\}$. Dobra gave a Markov basis for $\mathcal{F}^*_{\mathbf{n}_\mathcal{A}}$ consisting of only *degree-two* moves:

**Definition 3.** *Let $(A, S, B)$ be a partition of $V$. A degree-two move $\mathbf{z}$ has two positive entries and two negative entries:*

$$z(i, j, k) = 1, \quad z(i, j, k') = -1, \quad z(i', j, k) = -1, \quad z(i', j, k') = 1, \tag{4}$$

*where $i \neq i' \in \mathcal{X}_A$, $j \in \mathcal{X}_S$ $k \neq k', \in \mathcal{X}_B$. Let $\mathcal{M}^{d=2}(A, S, B)$ be the set of all degree-two moves generated from this partition.*

These are extensions of the well-known "swap moves" for two-dimensional contingency tables (e.g. [5]) to the subtable $\mathbf{n}(\cdot, j, \cdot)$, and they can be visualized as shown at right. In this arrangement, it is clear that any such move preserves the marginal table $\mathbf{n}_A$ (row sums) and the marginal table $\mathbf{n}_B$ (column sums); in other words, $\mathbf{z} \downarrow A = 0$ and $\mathbf{z} \downarrow B = 0$. Moreover, because $j$ is fixed, it is straightforward to see that $\mathbf{z} \downarrow A \cup S = 0$ and $\mathbf{z} \downarrow B \cup S = 0$. The cycle in Figure 1(e) is a degree-two move on the table $\mathbf{n}_{1,2}$, with $A = \{X_1\}, S = \emptyset, C = \{X_2\}$.

|  | $k$ | $k'$ |
|---|---|---|
| $i$ | $+$ | $-$ |
| $i'$ | $-$ | $+$ |

**Theorem 2** (Dobra [6]). *Let $\mathcal{A}$ be decomposable with $\bigcup \mathcal{A} = V$. Let $\mathcal{M}^*_\mathcal{A}$ be the union of the sets of degree-two moves $\mathcal{M}^{d=2}(A, S, B)$ where $S$ is a separator of $\mathcal{T}_\mathcal{A}$ and $(A, S, B)$ is the corresponding decomposition of $V$. Then $\mathcal{M}^*_\mathcal{A}$ is a Markov basis for $\mathcal{F}^*_{\mathbf{n}_\mathcal{A}}$.*

**Adaptation of Dobra basis to $\mathcal{F}_{\mathbf{n}_\mathcal{D}}$.** We now adapt the Dobra basis to our setting. Consider a complete table $\mathbf{n} \in \text{tbl}(V)$ and the configuration $\mathbf{n}_\mathcal{C} = \{\mathbf{n} \downarrow C\}_{C \in \mathcal{C}}$. Because marginalization is a linear operation, there is a linear operator $\mathbb{A}$ such that $\mathbf{n}_\mathcal{C} = \mathbb{A} \mathbf{n}_V$. Moreover, $\mathcal{F}_{\mathbf{n}_\mathcal{A}}$ is the image of $\mathcal{F}^*_{\mathbf{n}_\mathcal{A}}$ under $\mathbb{A}$. Thus, the image of the Dobra basis under $\mathbb{A}$ is a Markov basis for $\mathcal{F}_{\mathbf{n}_\mathcal{A}}$.

**Lemma 1.** *Let $\mathcal{M}^*_\mathcal{A}$ be a Markov basis for $\mathcal{F}^*_{\mathbf{n}_\mathcal{A}}$. Then $\mathcal{M}_\mathcal{A} = \{\mathbb{A}\mathbf{z} : \mathbf{z} \in \mathcal{M}^*_\mathcal{A}\}$ is a Markov basis for $\mathcal{F}_{\mathbf{n}_\mathcal{A}}$. We call $\mathcal{M}_\mathcal{A}$ the projected Dobra basis.*

*Proof.* Let $\mathbf{n}_\mathcal{C}, \mathbf{n}'_\mathcal{C} \in \mathcal{F}_{\mathbf{n}_\mathcal{A}}$. By consistency, there exist $\mathbf{n}_V, \mathbf{n}'_V \in \mathcal{F}^*_{\mathbf{n}_\mathcal{A}}$ such that $\mathbf{n}_\mathcal{C} = \mathbb{A}\mathbf{n}_V$ and $\mathbf{n}'_\mathcal{C} = \mathbb{A}\mathbf{n}'_V$. There is a sequence of moves $\mathbf{z}_1, \ldots, \mathbf{z}_L \in \mathcal{M}^*_\mathcal{A}$ leading from $\mathbf{n}'_V$ to $\mathbf{n}_V$, meaning that $\mathbf{n}'_V = \mathbf{n}_V + \sum_{\ell=1}^{L} \mathbf{z}_\ell$. By appliyng the linear operator $\mathbb{A}$ to both sides of this equation, we have that $\mathbf{n}'_\mathcal{C} = \mathbf{n}_\mathcal{C} + \sum_{\ell=1}^{L} \mathbb{A}\mathbf{z}_\ell$. Furthermore, each intermediate configuration $\mathbf{n}_\mathcal{C} + \sum_{\ell=1}^{L'} \mathbb{A}\mathbf{z}_\ell = \mathbb{A}(\mathbf{n}_V + \sum_{\ell=1}^{L'} \mathbf{z}_\ell) \in \mathcal{F}_{\mathbf{n}_\mathcal{A}}$. Thus $\mathcal{M}_\mathcal{A} = \{\mathbb{A}\mathbf{z} : \mathbf{z} \in \mathcal{M}^*_\mathcal{A}\}$ is a Markov basis for $\mathcal{F}_{\mathbf{n}_\mathcal{A}}$. $\square$

**Locality of moves.** First consider the case where all variables are part of some observed table, as in Dobra's setting. The practical message so far is that to sample from $p(\mathbf{n}_\mathcal{C} \mid \mathbf{n}_\mathcal{D})$, it suffices to generate moves from the projected Dobra basis $\mathcal{M}_\mathcal{D}$. This is done by first selecting a degree-two move $\mathbf{z} \in \mathcal{M}^*_\mathcal{D}$, and then marginalizing $\mathbf{z}$ onto each clique of $\mathcal{C}$. Naively, it appears that a single move may require us to update each clique. However, we will show that $\mathbf{z} \downarrow C$ will be zero for many cliques, a fact we can exploit to implement moves more efficiently. Let $(A, S, B)$ be the partition

used to generate $\mathbf{z}$. We deduce from the discussion following Definition 3 that $\mathbf{z} \downarrow C = 0$ unless $C$ has a nonempty intersection with both $A$ and $B$, so we may restrict our attention to these cliques, which form a connected subtree (Proposition S.1 in supplementary material). An implementation can then exploit this by pre-computing the connected subtrees for each separator and only generating the necessary components of the move. Algorithm 1 gives the details of generating moves.

**Unobserved variables.** Let us now consider settings where some variables are not part of any observed table, which may happen when the individual model has hidden variables, or, later, with noisy observations. Additional moves are needed to connect two configurations that disagree on marginal tables involving unobserved variables. Several approaches are possible. All require the introduction of *degree-one moves* $\mathbf{z} \in \mathcal{M}^{d=1}(A, B)$, which partition the variables into two sets $(A, B)$ and have two nonzero entries $z(i, j) = 1$, $z(i', j) = -1$ for $i \neq i' \in \mathcal{X}_A, j \in \mathcal{X}_B$. In the parlance of two-dimensional tables, these moves adjust two entries in a single column so they preserve the column sums ($\mathbf{n}_B$) but mod-

---

**Algorithm 1**: The projected Dobra basis $\mathcal{M}_A$

**Input**: Junction tree $\mathcal{T}_A$ with separators $\mathcal{S}_A$

1 *Before sampling*
2      For each $S \in \mathcal{S}_A$, find the associated decomposition $(A, S, B)$
3      Find the cliques $C \in \mathcal{C}$ that have non-empty intersection with both $A$ and $B$. These form a subtree of $\mathcal{T}_{\mathcal{C}}$. Denote these cliques by $\mathcal{C}_S$ and let $V_S = \bigcup \mathcal{C}_S$.
4      Let $A_S = A \cap V_S$ and $B_S = B \cap V_S$

5 *During sampling:* to generate a move for separator $S \in \mathcal{S}_A$
6      Select $\mathbf{z} \in \mathcal{M}^{d=2}(A_S, S, B_S)$
7      For each clique $C \in \mathcal{C}_S$, calculate $\mathbf{z} \downarrow C$

---

ify the row sums ($\mathbf{n}_A$). The cycle in Figure 1(d) is a degree-one move which adjusts the marginal table over $A = \{X_2\}$, but preserves the marginal table over $B = \{X_1, X_3\}$. We proceed once again by constructing a basis for complete tables and then marginalizing the moves onto cliques.

**Theorem 3.** *Let $\mathcal{U}$ be any decomposable collection on the set of unobserved variables $U = V \setminus \bigcup \mathcal{D}$, and let $\mathcal{D}' = \mathcal{D} \cup \mathcal{U}$. Let $\mathcal{M}^*$ consist of the moves $\mathcal{M}_{\mathcal{D}'}^*$ together with the moves $\mathcal{M}^{d=1}(A, V \setminus A)$ for each $A \in \mathcal{U}$. Then $\mathcal{M}^*$ is a Markov basis for $\mathcal{F}_{\mathbf{n}_\mathcal{D}}^*$, and $\mathcal{M} = \{\mathbb{A}\mathbf{z} : \mathbf{z} \in \mathcal{M}^*\}$ is a Markov basis for $\mathcal{F}_{\mathbf{n}_\mathcal{D}}$.*

Theorem 3 is proved in the supplementary material. The degree-one moves also become local upon marginalization: it is easy to check that $\mathbf{z} \downarrow C$ is zero unless $C \cap A$ is nonempty. These cliques also form a connected subtree. We recommend choosing $\mathcal{U}$ by restricting $\mathcal{T}_{\mathcal{C}}$ to the variables in $U$. This has the effect of adding degree-one moves for each clique of $\mathcal{C}$. By matching the structure of $\mathcal{T}_{\mathcal{C}}$, many of the additional degree-two moves become zero upon marginalization.

## 3.2 Constructing an efficient MCMC sampler

The second key challenge in constructing the MCMC sampler is utilizing the moves from the Markov basis in a way that *efficiently* explores the state space. A standard approach is to select a random move $\mathbf{z}$, a direction $\delta = \pm 1$ (each with probability $1/2$), and then propose the move $\mathbf{n}_{\mathcal{C}} + \delta \mathbf{z}$ in a Metropolis Hastings sampler. Although these moves are enough to connect any two configurations, we are particularly interested in problems where $M$ is large, for which moving by increments of $\pm 1$ will be prohibitively slow.

For general Markov bases, Diaconis and Sturmfels [5] suggest instead to construct a Gibbs sampler that uses the moves as directions for longer steps, by choosing the value of $\delta$ from the following distribution:

$$p(\delta) \propto p(\mathbf{n}_{\mathcal{C}} + \delta \mathbf{z} \mid \mathbf{n}_{\mathcal{D}}), \qquad \delta \in \{\delta : \mathbf{n}_{\mathcal{C}} + \delta \mathbf{z} \geq \mathbf{0}\}. \tag{5}$$

**Lemma 2** (Adapted from Diaconis and Sturmfels [5])**.** *Let $\mathcal{M}$ be a Markov basis for $\mathcal{F}_{\mathbf{n}_\mathcal{D}}$. Consider the Markov chain with moves $\delta \mathbf{z}$ generated by first choosing $\mathbf{z}$ uniformly at random from $\mathcal{M}$ and then choosing $\delta$ according to (5). This is a connected, reversible, aperiodic Markov chain on $\mathcal{F}_{\mathbf{n}_\mathcal{D}}$ with stationary distribution $p(\mathbf{n}_{\mathcal{C}} \mid \mathbf{n}_{\mathcal{D}})$.*

However, it is not obvious how to sample from $p(\delta)$. They suggest running a Markov chain in $\delta$, again having the property of moving in increments of one (see also [16]). In our case, the support of $p(\delta)$ may be as big as the population size $M$, so this solution remains unsatisfactory.

Fortunately, $p(\delta)$ has several properties that allow us to create a very efficient sampling algorithm. For a separator $S \in \mathcal{S}$, define $\mathbf{z}_S$ as $\mathbf{z}_C \downarrow S$ for any clique $C$ containing $S$. Now let $\mathcal{C}(\mathbf{z})$ be the

**Algorithm 2**: Sampling from $p(\delta)$ in constant time

**Input**: move $\mathbf{z}$ and current configuration $\mathbf{n}_\mathcal{C}$, with $|\mathcal{C}(\mathbf{z})| > 1$

1. Calculate $\delta_{\min}$ and $\delta_{\max}$ using (8)

2. Extend the function $f(\delta) := \log p(\delta)$ to the real line using the equality $n! = \Gamma(n+1)$ in Equation (7) for each constituent function $f_A(\delta) := \log p_A(\delta)$, $A \in \mathcal{S}(\mathbf{z}) \cup \mathcal{C}(\mathbf{z})$.

3. Use the logarithm of Equation (6) to evaluate $f(\delta)$ (for sampling) and its derivatives (for Newton's method):

$$f^{(q)}(\delta) = \sum_{C \in \mathcal{C}(\mathbf{z})} f_C^{(q)}(\delta) - \sum_{S \in \mathcal{S}(\mathbf{z})} f_S^{(q)}(\delta). \qquad q = 0, 1, 2.$$

Evaluate the derivatives of $f_A(\delta)$ using the logarithm of Equation (7) and the digamma and trigamma functions $\psi(n) = \frac{d}{dn}\Gamma(n)$ and $\psi_1(n) = \frac{d^2}{dn^2}\Gamma(n)$.

4. Find the mode $\delta^*$ by first using Newton's method to find $\delta'$ maximizing $f(\delta)$ over the real line, and then letting $\delta^*$ be the value in $\{\lfloor \delta' \rfloor, \lceil \delta' \rceil, \delta_{\min}, \delta_{\max}\}$ that attains the maximum.

5. Run the rejection sampling algorithm of Devroye [3].

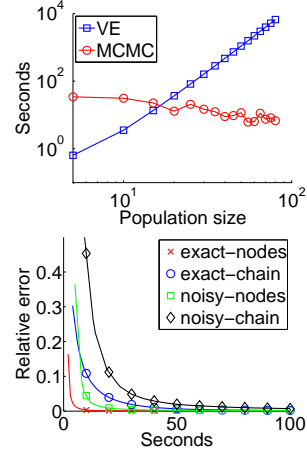

Figure 2: Top: running time vs. $M$ for a small CGM. Bottom: convergence of MCMC for random Bayes nets.

set of cliques $C$ for which $\mathbf{z}_C$ is nonzero, and let $\mathcal{S}(\mathbf{z})$ be defined analogously. For $A \in \mathcal{S} \cup \mathcal{C}$, let $\mathcal{I}^+(\mathbf{z}_A) \subseteq \mathcal{X}_A$ be the indices of $+1$ entries of $\mathbf{z}_A$ and let $\mathcal{I}^-(\mathbf{z}_A)$ be the indices of $-1$ entries. By ignoring constant terms in (2), we can write (5) as

$$p(\delta) \propto \prod_{C \in \mathcal{C}(\mathbf{z})} p_C(\delta) \prod_{S \in \mathcal{S}(\mathbf{z})} p_S(\delta)^{-1}, \qquad (6)$$

$$p_A(\delta) := \prod_{i \in \mathcal{I}^+(\mathbf{z}_A)} \frac{\mu_A(i)^\delta}{(n_A(i) + \delta)!} \prod_{j \in \mathcal{I}^-(\mathbf{z}_A)} \frac{\mu_A(j)^{-\delta}}{(n_A(j) - \delta)!}, \qquad A \in \mathcal{S} \cup \mathcal{C}. \qquad (7)$$

To maintain the non-negativity of $\mathbf{n}_\mathcal{C}$, $\delta$ is restricted to the support $\delta_{\min}, \ldots, \delta_{\max}$ with:

$$\delta_{\min} := - \min_{C \in \mathcal{C}(\mathbf{z}), i \in \mathcal{I}^+(\mathbf{z}_C)} n_C(i), \qquad \delta_{\max} := \min_{C \in \mathcal{C}(\mathbf{z}), j \in \mathcal{I}^-(\mathbf{z}_C)} n_C(j). \qquad (8)$$

Notably, each move in our basis satisfies $|\mathcal{I}^+(\mathbf{z}_A) \cup \mathcal{I}^+(\mathbf{z}_A)| \leq 4$, so $p(\delta)$ can be evaluated by examining at most four entries in each table for cliques in $\mathcal{C}(\mathbf{z})$. It is worth noting that Equation (7) reduces to the binomial distribution for degree-one moves and the (noncentral) hypergeometric distribution for degree-two moves, so we may sample from these distributions directly when $|\mathcal{C}(\mathbf{z})| = 1$. More importantly, we will now show that $p(\delta)$ is always a member of the *log-concave* class of distributions, which are unimodal and can be sampled very efficiently.

**Definition 4.** *A discrete distribution $\{p_k\}$ is* log-concave *if $p_k^2 \geq p_{k-1}p_{k+1}$ for all $k$ [3].*

**Theorem 4.** *For any degree-one or degree-two move $\mathbf{z}$, the distribution $p(\delta)$ is log-concave.*

It is easy to show that both $p_C(\delta)$ and $p_S(\delta)$ are log-concave. The proof of Theorem 4, which is found in the supplementary material, then pairs each separator $S$ with a clique $C$ and uses properties of the moves to show that $p_C(\delta)/p_S(\delta)$ is also log-concave. Then, by Equation (6), we see that $p(\delta)$ is a product of log-concave distributions, which is also log-concave.

We have implemented the rejection sampling algorithm of Devroye [3], which applies to any discrete log-concave distribution and is simple to implement. The expected number of times it evaluates $p(\delta)$ (up to normalization) is fewer than 5. We must also provide the mode of the distribution, which we find by Newton's method, usually taking only a few steps. The running time for each move is thus *independent of the population size*. Additional details are given in Algorithm 2.

### 3.3 Noisy Observations

Population-level counts from real survey data are rarely exact, and it is thus important to incorporate noisy observations into our model. In this section, we describe how to modify the sampler for

the case when all observations are noisy; it is a straightforward generalization to allow both noisy and exact observations. Suppose that we make noisy observations $\mathbf{y}_\mathcal{R} = \{\mathbf{y}_R : R \in \mathcal{R}\}$ corresponding to the *true* marginal tables $\mathbf{n}_\mathcal{R}$ for a collection $\mathcal{R} \preceq \mathcal{C}$ (that need *not* be decomposable). For simplicity, we restrict our attention to models where each entry $n$ in the true table is corrupted independently according to a univariate noise model $p(y \mid n)$.

We assume that the noise model is log-concave, meaning in this case that $\log p(y \mid n)$ is a concave function of the *parameter* $n$. Most commonly-used univariate densities are log-concave with respect to various parameters [17]. A canonical example from the bird migration model is $p(y \mid n) =$ Poisson$(\alpha n)$, so the survey count is Poisson with mean proportional to the true number of birds present. This example and others are discussed in [2]. We also assume that the support of $p(y \mid n)$ does not depend on $n$, so that observations do not restrict the support of the sampling distribution. For example, we must modify our Poisson noise model to be $p(y \mid n) = $ Poisson$(\alpha n + \lambda_0)$ with small background rate $\lambda_0$ to avoid the hard constraint that $n$ must be positive if $y$ is positive.

In analogy with (3), we can then write $p(\mathbf{n}_\mathcal{C} \mid \mathbf{y}_\mathcal{R}) \propto p(\mathbf{n}_\mathcal{C})p(\mathbf{y}_\mathcal{R} \mid \mathbf{n}_\mathcal{C})$ (the hard constraint is now replaced with the likelihood term $p(\mathbf{y}_\mathcal{R} \mid \mathbf{n}_\mathcal{C})$). Given our assumption on $p(y \mid n)$, the support of $p(\mathbf{n}_\mathcal{C} \mid \mathbf{y}_\mathcal{R})$ is the same as the support of $p(\mathbf{n}_\mathcal{C})$, and a Markov basis can be constructed using the tools from Section 3.1, with all variables being unobserved. In the sampler, the expression for $p(\delta)$ must now be updated to incorporate the likelihood term $p(\mathbf{y}_\mathcal{R} \mid \mathbf{n}_\mathcal{C} + \delta\mathbf{z})$. Following reasoning similar to before, we let $\mathcal{R}(\mathbf{z})$ be the sets in $\mathcal{R}$ for which $\mathbf{z} \downarrow R$ is nonzero and find that Equation (6) gains the additional factor $\prod_{R \in \mathcal{R}(\mathbf{z})} p_R(\delta)$, where

$$p_R(\delta) = \prod_{i \in \mathcal{I}^+(\mathbf{z}_R)} p(y_R(i) \mid n_R(i) + \delta) \prod_{j \in \mathcal{I}^-(\mathbf{z}_R)} p(y_R(j) \mid n_R(j) - \delta). \tag{9}$$

Each factor in (9) is log-concave in $\delta$ by our assumption on $p(y \mid n)$, and hence the overall distribution $p(\delta)$ remains log-concave. To update the sampler for $p(\delta)$, modify line 3 of Algorithm 2 in the obvious fashion to include these new factors when computing $\log p(\delta)$ and its derivatives.

## 4  Experiments

We implemented our sampler in MATLAB using Murphy's Bayes net toolbox [18] for the underlying operations on graphical models and junction trees. Figure 2 (top) compares the running time of our method vs. exact inference in the CGM by variable elimination (VE) for a very small model. The task was to estimate $E[\mathbf{n}_{2,3} \mid \mathbf{n}_1, \mathbf{n}_3]$ in the bird migration model for $L = 2, T = 3$, and varying $M$. The running time of VE is $O(M^{L^2-1})$, which is cubic in $M$ (linear on a log-log plot), while the time for our method to estimate the same quantity within 2% relative error actually decreases slightly with population size. Figure 2 (bottom) shows convergence of the sampler for more complex models. We generated 30 random Bayes nets on 10 binary variables, and generated two sets of observed tables for a population of $M = 100,000$: the set NODES has a table for each single variable, while the set CHAIN has tables for pairs of variables that are adjacent in a random ordering. We repeated the same process with the noise model $p(y \mid n) = $ Poisson$(0.2n + 0.1)$ to generate noisy observations. We then ran our sampler to estimate $E[\mathbf{n}_\mathcal{C} \mid \mathbf{n}_\mathcal{D}]$ as would be done in the EM algorithm. The plots show relative error in this estimate as a function of time, averaged over the 30 nets. For more details, including how we derived the correct answer for comparison, see Section 4.1 in the supplementary material. The sampler converged quickly in all cases with the more complex CHAIN observation model taking longer than NODES, and noisy observations taking slightly longer than exact ones. We found (not shown) that the biggest source of variability in convergence time was due to individual Bayes nets, while repeat trials using the same net demonstrated very similar behavior.

**Concluding Remarks.** An important area of future research is to further explore the use of CGMs within learning algorithms, as well as the limitations of that approach: when is it possible to learn individual models from aggregate data? We believe that the ability to model noisy observations will be an indispensable tool in real applications. For complex models, convergence may be difficult to diagnose. Some mixing results are known for samplers in related problems with hard constraints [16]; any such results for our model would be a great advance. The use of distributional approximations for the CGM model and other methods of approximate inference also hold promise.

**Acknowledgments.** We thank Lise Getoor for pointing out the connection between CGMs and lifted inference. This research was supported in part by the grant DBI-0905885 from the NSF.

# References

[1] D. Sheldon, M. A. S. Elmohamed, and D. Kozen. Collective inference on Markov models for modeling bird migration. In *Advances in Neural Information Processing Systems (NIPS 2007)*, pages 1321–1328, Cambridge, MA, 2008. MIT Press.

[2] Daniel Sheldon. *Manipulation of PageRank and Collective Hidden Markov Models*. PhD thesis, Cornell University, 2009.

[3] L. Devroye. A simple generator for discrete log-concave distributions. *Computing*, 39(1): 87–91, 1987.

[4] A. Agresti. A survey of exact inference for contingency tables. *Statistical Science*, 7(1):131–153, 1992.

[5] P. Diaconis and B. Sturmfels. Algebraic algorithms for sampling from conditional distributions. *The Annals of statistics*, 26(1):363–397, 1998. ISSN 0090-5364.

[6] A. Dobra. Markov bases for decomposable graphical models. *Bernoulli*, 9(6):1093–1108, 2003. ISSN 1350-7265.

[7] S.L. Lauritzen. *Graphical models*. Oxford University Press, USA, 1996.

[8] D. Poole. First-order probabilistic inference. In *Proc. IJCAI*, volume 18, pages 985–991, 2003.

[9] R. de Salvo Braz, E. Amir, and D. Roth. Lifted first-order probabilistic inference. *Introduction to Statistical Relational Learning*, page 433, 2007.

[10] B. Milch, L.S. Zettlemoyer, K. Kersting, M. Haimes, and L.P. Kaelbling. Lifted probabilistic inference with counting formulas. *Proc. 23rd AAAI*, pages 1062–1068, 2008.

[11] P. Sen, A. Deshpande, and L. Getoor. Bisimulation-based approximate lifted inference. In *Proceedings of the Twenty-Fifth Conference on Uncertainty in Artificial Intelligence*, pages 496–505. AUAI Press, 2009.

[12] J. Kisynski and D. Poole. Lifted aggregation in directed first-order probabilistic models. In *Proc. IJCAI*, volume 9, pages 1922–1929, 2009.

[13] Udi Apsel and Ronen Brafman. Extended lifted inference with joint formulas. In *Proceedings of the Proceedings of the Twenty-Seventh Conference Annual Conference on Uncertainty in Artificial Intelligence (UAI-11)*, pages 11–18, Corvallis, Oregon, 2011. AUAI Press.

[14] R. Sundberg. Some results about decomposable (or Markov-type) models for multidimensional contingency tables: distribution of marginals and partitioning of tests. *Scandinavian Journal of Statistics*, 2(2):71–79, 1975.

[15] M.J. Wainwright and M.I. Jordan. Graphical models, exponential families, and variational inference. *Foundations and Trends in Machine Learning*, 1(1-2):1–305, 2008.

[16] P. Diaconis, S. Holmes, and R.M. Neal. Analysis of a nonreversible Markov chain sampler. *The Annals of Applied Probability*, 10(3):726–752, 2000.

[17] W.R. Gilks and P. Wild. Adaptive Rejection sampling for Gibbs Sampling. *Journal of the Royal Statistical Society. Series C (Applied Statistics)*, 41(2):337–348, 1992. ISSN 0035-9254.

[18] K. Murphy. The Bayes net toolbox for MATLAB. *Computing science and statistics*, 33(2): 1024–1034, 2001.

